# HMM Speech Recognition
# with Neural Net Discrimination*

**William Y. Huang and Richard P. Lippmann**
Lincoln Laboratory, MIT
Room B–349
Lexington, MA 02173–9108

## ABSTRACT

Two approaches were explored which integrate neural net classifiers with Hidden Markov Model (HMM) speech recognizers. Both attempt to improve speech pattern discrimination while retaining the temporal processing advantages of HMMs. One approach used neural nets to provide second–stage discrimination following an HMM recognizer. On a small vocabulary task, Radial Basis Function (RBF) and back–propagation neural nets reduced the error rate substantially (from 7.9% to 4.2% for the RBF classifier). In a larger vocabulary task, neural net classifiers did not reduce the error rate. They, however, outperformed Gaussian, Gaussian mixture, and $k$–nearest neighbor (KNN) classifiers. In another approach, neural nets functioned as low–level acoustic–phonetic feature extractors. When classifying phonemes based on single 10 msec. frames, discriminant RBF neural net classifiers outperformed Gaussian mixture classifiers. Performance, however, differed little when classifying phones by accumulating scores across all frames in phonetic segments using a single node HMM recognizer.

*This work was sponsored by the Department of the Air Force and the Air Force Office of Scientific Research.

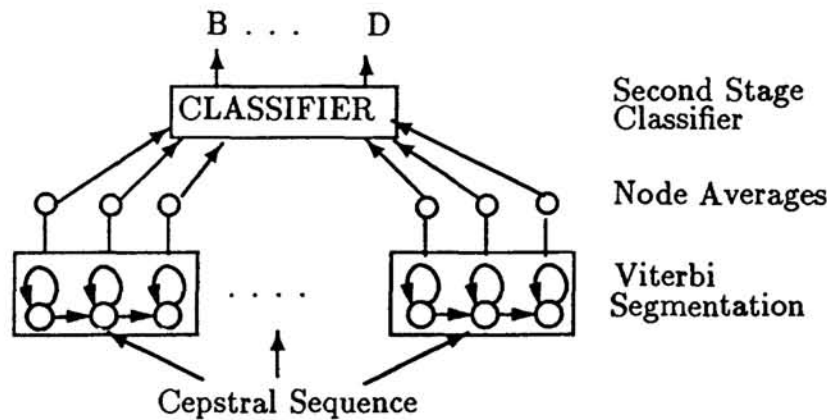

**Figure 1:** Second stage discrimination system. HMM recognition is based on the accumulated scores from each node. A second stage classifier can adjust the weights from each node to provide improved discrimination.

## 1   Introduction

This paper describes some of our current efforts to integrate discriminant neural net classifiers into HMM speech recognizers. The goal of this work is to combine the temporal processing capabilities of the HMM approach with the superior recognition rates provided by discriminant classifiers. Although neural nets are well developed for static pattern classification, neural nets for dynamic pattern recognition require further research. Current conventional HMM recognizers rely on likelihood scores provided by non–discriminant classifiers, such as Gaussian mixture [11] and histogram [5] classifiers. Non–discriminant classifiers are sensitive to assumptions concerning the shape of the probability density function and the robustness of the Maximum Likelihood (ML) estimators. Discriminant classifiers have a number of potential advantages over non–discriminant classifiers on real world problems. They make fewer assumptions concerning underlying class distributions, can be robust to outliers, and can lead to efficient parallel analog VLSI implementation [4, 6, 7, 8]. Recent efforts in applying discriminant training to HMM recognizers have led to promising techniques, including Maximum Mutual Information (MMI) training [2] and corrective training [5]. These techniques maintain the same structure as in a conventional HMM recognizer but use a different overall error criteria to estimate parameters. We believe that a significant improvement in recognition rate will result if discriminant classifiers are included directly in the HMM structure.

This paper examines two integration strategies: second stage classification and discriminant pre-processing. In second stage classification, discussed in Sec. 2, classifiers are used to provide post-processing for an HMM isolated word recognizer. In discriminant pre-processing, discussed in Sec. 3, discriminant classifiers replace the maximum likelihood classifiers used in conventional HMM recognizers.

## 2    Second Stage Classification

HMM isolated–word recognition requires one Markov model per word. Recognition involves accumulating scores for an unknown input across the nodes in each word model, and selecting that word model which provides the maximum accumulated score. In the case of discriminating between minimal pairs, such as those in the E–set vocabulary (the letters {BCDEGPTVZ}), it is desired that recognition be focused on the nodes that correspond to the small portion of the utterance that are different between words. In the second stage classification approach, illustrated in Fig. 1, the HMMs at the first layer are the components of a fully–trained isolated–word HMM recognizer. The second stage classifier is provided with matching scores and duration from each HMM node. A simple second stage classifier which sums the matching scores of the nodes for each word would be equivalent to an HMM recognizer. It is hoped that discriminant classifiers can utilize the additional information provided by the node dependent scores and duration to deliver improved recognition rates.

The second stage system of Fig. 1 was evaluated using the 9 letter E-set vocabulary and the {BDG} vocabulary. Words were taken from the TI–46 Word database, which contains 10 training and 16 testing tokens per word per talker and 16 talkers. Evaluation was performed in the speaker dependent mode; thus, there were a total of 30 training and 48 testing tokens per talker for the {BDG}-set task and 90 training and 144 testing tokens per talker for the E-set task. Spectral pre–processing consisted of extracting the first 12 mel–scaled cepstral coefficients [10], ignoring the $0^{\text{th}}$ cepstral coefficient (energy), for each 10 ms frame. An HMM isolated word recognizer was first trained using the forward–backward algorithm. Each word was modeled using 8 HMM nodes with 2 additional noise nodes at each end. During classification, each test word was segmented using the Viterbi decoding algorithm on all word models. The average matching score and duration of all non–noise nodes were used as a static pattern for the second stage classifier.

### 2.1    Classifiers

Four second stage classifiers were used: (1) Multi–layer perceptron (MLP) classifiers trained with back-propagation, (2) Gaussian mixture classifiers trained with the Expectation Maximization (EM) algorithm [9], (3) RBF classifiers [8] with weights trained using the pseudoinverse method computed via Singular Value Decomposition (SVD), and (4) KNN classifiers. Covariance matrices in the Gaussian mixture classifiers were constrained to be diagonal and tied to be the same between mixture components in all classes. The RBF classifiers were of the form

$$\text{Decide Class } i = \underset{i}{\text{Argmax}} \sum_{j=1}^{J} w_{ij} \text{EXP} \left( -\frac{\|\vec{x} - \vec{\mu}_j\|^2}{2h\sigma_j^2} \right) \tag{1}$$

where

$$
\begin{aligned}
\vec{x} &= \text{acoustic vector input,} \\
i &= \text{class label,} \\
J &= \text{number of centers,} \\
w_{ij} &= \text{weight from } j^{\text{th}} \text{ center to } i^{\text{th}} \text{ class output,} \\
(\vec{\mu}_j, \sigma_j^2) &= j^{\text{th}} \text{ center and variance, and} \\
h &= \text{spread factor.}
\end{aligned}
$$

The center locations ($\vec{\mu}_i$'s) were obtained from either $k$-means or Gaussian mixture clustering. The variances ($\sigma_j$'s) were either the variances of the individual $k$-means clusters or those of the individual Gaussian mixture components, depending on which clustering algorithm was used. Results for $k = 1$ are reported for the KNN classifier because this provided best performance.

The Gaussian mixture classifier was selected as a reference conventional non–discriminant classifier. A Gaussian mixture classifier can provide good models for multi-modal and non–Gaussian distributions by using many mixture components. It can also generalize to the more common, well–known unimodal Gaussian classifier which provides poor performance when the input distribution is not Gaussian. Very few benchmarking studies have been performed to evaluate the relative performance of Gaussian mixture and neural net classifiers, although mixture models have been used successfully in HMM recognizers [11]. RBF classifiers were used because they train rapidly, and recent benchmarking studies show that they perform as well as MLP classifiers on speech problems [8].

| | | | GAUSSIAN | | RBF† | | RBF‡ | | |
| | | | Mixtures per Class | | Centers per Class | | Total Number of Centers | | KNN |
| Vocab | HMM | MLP | 1 | 3 | 1 | 3 | 30 | 70 | ($k = 1$) |
|---|---|---|---|---|---|---|---|---|---|
| {BDG} | 7.9% | 5.9% | 5.6% | 9.1% | 11.9% | 5.7% | 4.2% | | 6.0% |
| {E–Set} | 11.3% | 13.4% | 21.2% | 20.6% | 15.8% | 13.7% | 15.8% | 12.8% | 36.0% |

† Centers from Gaussian mixture clustering, h=150.
‡ Centers from $k$-means clustering. h=150.

Table 1: Percentage errors from the second stage classifier, averaged over all 16 talkers.

## 2.2   Results of Second Stage Classification

Table 1 shows the error rates for the second stage system of Fig. 1, averaged over all talkers. The second stage system improved performance over the baseline HMM system when the vocabulary was small (B, D and G). Error rates decreased from 7.9% for the baseline HMM recognizer to 4.2% for the RBF second stage classifier. There was no improvement for the E–set vocabulary task. The best RBF second stage classifier degraded the error rate from 11.3% with the baseline HMM to 12.8%. In the E-set results, MLP and RBF classifiers, with error rates of 13.4%

and 12.8%, performed considerably better than the Gaussian (21.2%), Gaussian mixture (20.6%) and KNN classifiers (36.0%).

The second stage approach is effective for a very small vocabulary but not for a larger vocabulary task. This may be due to a combination of limited training data and the increased complexity of decision regions as vocabulary size and dimensionality gets large. When the vocabulary size increased from 3 to 9, the input dimensionality of the classifiers scaled up by a factor of 3 (from 48 to 144) but the number of training tokens increased only by the same factor (from 30 to 90). It is, in general, possible for the amount of training tokens required for good performance to scale up exponentially with the input dimensionality. MLP and RBF classifiers appear to be affected by this problem but not as strongly as Gaussian, Gaussian mixture, and KNN classifiers.

## 3    Discriminant Pre–Processing

Second stage classifiers will not work well if the nodal matching scores do not lead to good discrimination. Current conventional HMM recognizers use non–discriminant classifiers based on ML estimators to generate these scores. In the discriminant pre–processing approach, the ML classifiers in an HMM recognizer are replaced by discriminant classifiers.

All the experiments in this section are based on the phonemes /b,d,ʤ/ from the speaker dependent TI–46 Word database. Spectral pre–processing consisted of extracting the first 12 mel–scaled cepstral coefficients and ignoring the $0^{th}$ cepstral coefficient (energy), for each 10 ms frame. For multi–frame inputs, adjacent frames were 20 msec. apart (skipping every other frame). The database was segmented with a conventional high–performance continuous–observation HMM recognizer using forced Viterbi decoding on the correct word. The phonemes /b/, /d/ and /ʤ/ from the letters "B", "D" and "G" (/#_i/ context) were then extracted. This resulted in an average of 95 training and 158 testing frames per talker per word using the 10 training and 16 testing words per talker in the 16 talker database. Talker dependent results, averaged over all 16 talkers, are reported here.

Preliminary experiments using MLP, RBF, KNN, Gaussian, and Gaussian mixture classifiers indicated that RBF classifiers with Gaussian basis functions and a spread factor of 50 consistently yielded close to best performance. RBF classifiers also provided much shorter training times than MLP classifiers. RBF classifiers (as in Eq. 1) with $h = 50$ were thus used in all experiments presented in this section. The parameters of the RBF classifiers were determined as described in Sec. 2.1 above.

Gaussian mixture classifiers were used as reference conventional non–discriminant classifiers. In the preliminary experiments, they also provided close to best performance, and outperformed KNN and unimodal Gaussian classifiers. Covariance matrices were constrained, as described in Sec. 2.1. Although full and independent covariance matrices were advantageous for the unimodal Gaussian classifier and Gaussian mixture classifiers with few mixture components, best performance was provided using many mixture components and constrained covariance matri-

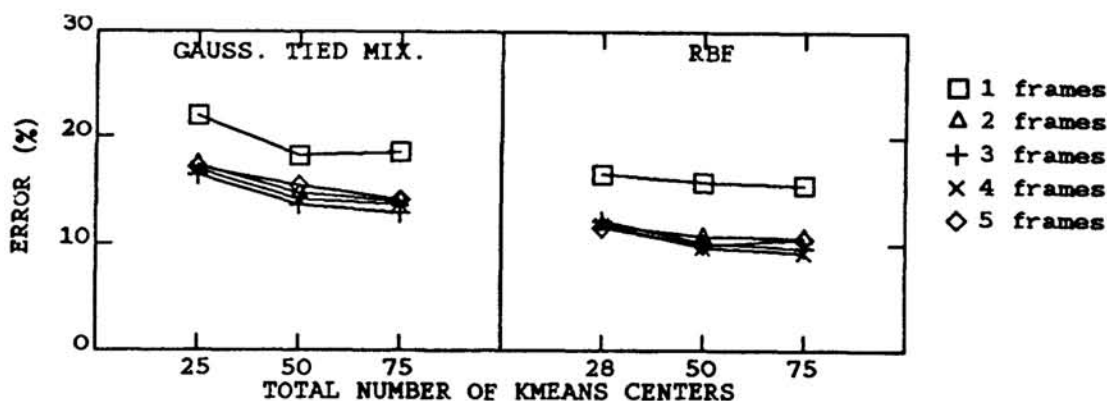

**Figure 2:** Frame–level error rates for Gaussian tied–mixture and RBF classifiers as a function of the total number of unique centers. Multi–frame results had context frames adjoined together at the input. Centers for both classifiers were determined using $k$–means clustering.

ces. A Gaussian "tied–mixture" classifier was also used. This is a Gaussian mixture classifier where all classes share the same mixture components but have different mixture weights. It is trained in two stages. In the first stage, class independent mixture centers are computed by $k$–means clustering, and mixture variances are the variances of the individual $k$–means clusters. In the second stage, the ML estimates of the class dependent mixture weights are computed while holding mixture components fixed.

## 3.1  Frame Level Results

Error rates for classifying phonemes based on single frames are shown in Fig. 2 for the Gaussian tied–mixture classifier (left) and RBF classifier (right). These results were obtained using $k$-means centers. Superior frame–level error rates were consistently provided by the RBF classifier in all experimental variations of this study. This is expected since RBF classifiers use an objective function which is directly related to classification error, whereas the objective of non–discriminant classifiers, modeling the class dependent probability density functions, is only indirectly related to classification error.

## 3.2  Phone Level Results

In a single node HMM, classifier scores for the frames in a phone segment are accumulated to obtain phone–level results. For conventional HMM recognizers that use non–discriminant classifiers, this score accumulation is done by assuming independent frames, which allows the frame–level scores to be multiplied together:

$$
\begin{aligned}
Prob(\text{phone}) &= Prob(\vec{x}_1, \vec{x}_2, ...\vec{x}_N) \\
&= Prob(\vec{x}_1)Prob(\vec{x}_2)\cdots Prob(\vec{x}_N)
\end{aligned}
\tag{2}
$$

where $\vec{x}...\vec{x}_N$ are input frames in an $N$–frame phone. Eq. 2 does not apply to non–discriminant classifiers. RBF classifier outputs are not constrained to lie between 0 and 1. They do not necessarily behave like probabilities and do not perform

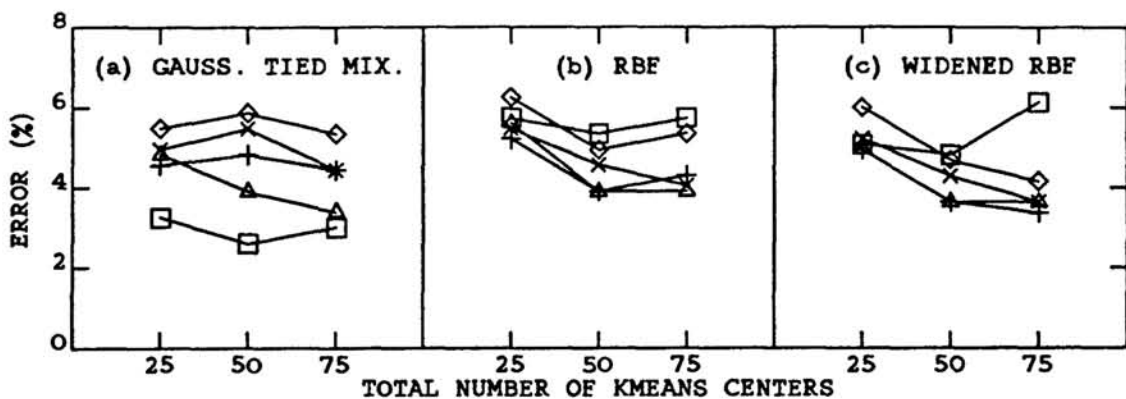

**Figure 3:** Phone–level error rates using (a) Gaussian tied–mixture, (b) RBF and (c) 5% widened RBF classifiers, as a function of the total number of unique centers. Gaussian classifier phone–level results were obtained by accumulating frame–level scores via multiplication. RBF classifier frame–level scores were accumulated via addition. Symbols are as in Fig. 2.

well when their frame scores are multiplied together. The RBF classifier's frame–level scores were thus accumulated, instead, by addition. Phone–level error rates obtained by accumulating frame–level scores from the Gaussian tied–mixture and RBF classifiers are shown in Fig.'s 3(a) and (b). Best performance was provided by the Gaussian tied–mixture classifier with 50 $k$–means centers and no context frames (2.6% error rate, versus 3.9% for the RBF classifier with 75 centers and 1 context frame).

The good phone–level performance provided by the Gaussian tied–mixture classifier in Fig. 3(a) is partly due to the near correctness of the Gaussian mixture distribution assumption and the independent frames assumption (Eq. 2). To address the poor phone–level performance of the RBF classifier, we examine solutions that use smoothing to directly extend good frame–level results to acceptable phone–level performance. Smoothing was performed both by passing the classifier outputs through a sigmoid function[1] and by increasing the spread ($h$ in Eq. 1) *after* RBF weights were trained. Increasing $h$ was more effective.

Increasing $h$ has the effect of "widening" the basis functions. This smoothes the discriminant functions produced by the RBF classifier to compensate for limited training data. If basis function widening occurs before weights are trained, then weights training will effectively compensate for the increase. This was verified in preliminary experiments, which showed that if $h$ was increased before weights were trained, little difference in performance was observed as $h$ varies from 50 to 200. Increasing $h$ by 5% after weights were trained resulted in a slightly different frame–level performance (sometimes better, sometimes worse), but a significant improvement in phone–level results for all experimental variations of this study. In Fig. 3(c), a 5% widening of the basis function improved the performance of the baseline

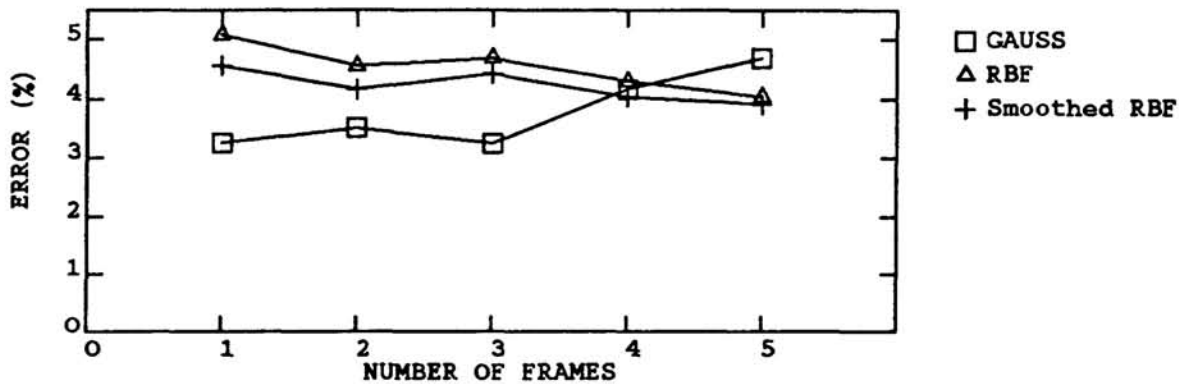

**Figure 4:** Phone–level error rates, as a function of the number of frames, for Gaussian mixture with 9 mixtures per class, and RBF classifiers with centers from the Gaussian mixture classifier (27 total centers for this 3 class task).

RBF classifier. It did not, however, improve performance over that provided by the Gaussian tied–mixture classifier without context frames at the input. The lowest error rate provided by the smoothed RBF is now 3.4% using 75 $k$–means centers and 2 context frames (compared with 2.6% for the Gaussian tied–mixture classifier with 50 centers and no context).

Error rates for the Gaussian mixture classifier with 9 mixtures per class is plotted versus the number of frames in Fig. 4, along with the results for RBF classifiers with centers taken from the Gaussian mixture classifier. Similar behavior was observed in all experimental variations of this study. There are three main observations: (1) The Gaussian mixture classifier without context frames provided best performance but degraded as the number of input frames increased, (2) RBF classifiers can outperform Gaussian mixture classifiers with many input frames, and (3) widening the basis functions after weights were trained improved the RBF classifier's performance.

## 4  Summary

Two techniques were explored that integrated discriminant classifiers into HMM speech recognizers. In second–stage discrimination, an RBF second–stage classifier halved the error rates in a {BDG} vocabulary task but provided no performance improvement in an E-set vocabulary task. For integrating at the pre–processing level, RBF classifiers provided superior frame–level performance over conventional Gaussian mixture classifiers. At the phone–level, best performance was provided by a Gaussian mixture classifier with a single frame input; however, the RBF classifier outperformed the Gaussian mixture classifier when the input contained multiple context frames. Both sets of experiments indicated an ability for the RBF classifier to integrate the large amount of information provided by inputs with high dimensionality. They suggest that an HMM recognizer integrated with RBF and other discriminant classifiers may provide improved recognition by providing better frame–level discrimination and by utilizing features that are ignored by current "state–of–the–art" HMM speech recognizers. This is consistent with the results of

Franzini [3] and Bourlard [1], who used many context frames in their implementation of discriminant pre–processing which embedded MLPs' into HMM recognizers.

Current efforts focus on studying techniques to improve the performance of discriminant classifier for phones, words, and continuous speech. Approaches include accumulating scores from lower level speech units and using objective functions that depend on higher level speech units, such as phones and words. Work is also being performed to integrate discriminant classification algorithms into HMM recognizers using Viterbi training.

## Footnotes

[1] The sigmoid function is of the form $y = 1/\left(1 + e^{-(x-.5)^2}\right)$ where $x$ is the input (an output from the RBF classifier) and $y$ is the output used for classification.

# References

[1] H. Bourlard and N. Morgan. Merging multilayer perceptrons in hidden Markov models: Some experiments in continuous speech recognition. Technical Report TR–89–033, International Computer Science Institute, Berkeley, CA., July 1989.

[2] Peter F. Brown. *The Acoustic-Modeling Problem in Automatic Speech Recognition.* PhD thesis, Carnegie Mellon University, May 1987.

[3] Michael A. Franzini, Michael J. Witbrock, and Kai-Fu Lee. A connectionist approach to continuous speech recognition. In *Proceedings of the IEEE ICASSP*, May 1989.

[4] William Y. Huang and Richard P. Lippmann. Comparisons between conventional and neural net classifiers. In *1st International Conference on Neural Network*, pages IV–485. IEEE, June 1987.

[5] Kai-Fu Lee and Sanjoy Mahajan. Corrective and reinforcement leaning for speaker–independent continuous speech recognition. Technical Report CMU-CS-89-100, Computer Science Department, Carnegie–Mellon University, January 1989.

[6] Yuchun Lee and Richard Lippmann. Practical characteristics of neural network and conventional pattern classifiers on artificial and speech problems. In *Advances in Neural Information Processing Systems 2*, Denver, CO., 1989. IEEE, Morgan Kaufmann. In Press.

[7] R. P. Lippmann. Review of neural networks for speech recognition. *Neural Computation*, 1(1):1–38, 1989.

[8] Richard P. Lippmann. Pattern classification using neural networks. *IEEE Communications Magazine*, 27(11):47–63, Nov. 1989.

[9] G. J. McLachlan. *Mixture Models.* Marcel Dekker, New York, N. Y., 1988.

[10] D. B. Paul. A speaker-stress resistant HMM isolated word recognizer. In *Proceedings of the IEEE ICASSP*, pages 713–716, April 1987.

[11] L. R. Rabiner, B.-H. Juang, S. E. Levinson, and M. M. Sondhi. Recognition of isolated digits using hidden Markov models with continuous mixture densities. *AT&T Technical Journal*, 64(6):1211–1233, 1985.
